# Single-iteration Threshold Hamming Networks

Isaac Meilijson              Eytan Ruppin

Moshe Sipper
School of Mathematical Sciences
Raymond and Beverly Sackler Faculty of Exact Sciences
Tel Aviv University, 69978 Tel Aviv, Israel

## Abstract

We analyze in detail the performance of a Hamming network classifying inputs that are distorted versions of one of its $m$ stored memory patterns. The activation function of the memory neurons in the original Hamming network is replaced by a simple threshold function. The resulting Threshold Hamming Network (THN) correctly classifies the input pattern, with probability approaching 1, using only $O(m \ln m)$ connections, in a single iteration. The THN drastically reduces the time and space complexity of Hamming Network classifiers.

## 1   Introduction

Originally presented in (Steinbuch 1961, Taylor 1964) the Hamming network (HN) has received renewed attention in recent years (Lippmann et. al. 1987, Baum et. al. 1988). The HN calculates the Hamming distance between the input pattern and each memory pattern, and selects the memory with the smallest distance. It is composed of two subnets: The *similarity* subnet, consisting of an $n$-neuron input layer connected with an $m$-neuron memory layer, calculates the number of equal bits between the input and each memory pattern. The *winner-take-all* (WTA) subnet, consisting of a fully connected $m$-neuron topology, selects the memory neuron that best matches the input pattern.

The similarity subnet uses $mn$ connections and performs a single iteration. The WTA subnet has $m^2$ connections. With randomly generated input and memory patterns, it converges in $\Theta(m \ln(mn))$ iterations (Floreen 1991). Since $m$ is exponential in $n$, the space and time complexity of the network is primarily due to the WTA subnet (Domany & Orland 1987). We analyze the performance of the HN in the practical scenario where the input pattern is a distorted version of some stored memory vector. We show that it is possible to replace the original activation function of the neurons in the memory layer by a simple threshold function, and completely discard the WTA subnet. If the threshold is properly tuned, only the neuron standing for the 'correct' memory is likely to be activated. The resulting Threshold Hamming Network (THN) will perform correctly (with probability approaching 1) in a single iteration, using only $O(m \ln m)$ connections instead of the $O(m^2)$ connections in the original HN. We identify the optimal threshold, and measure its performance relative to the original HN.

## 2    The Threshold Hamming Network

We examine a HN storing $m + 1$ memory patterns $\xi^\mu$, $1 \leq \mu \leq m + 1$, each being an $n$-dimensional vector of $\pm 1$. The input pattern $x$ is generated by selecting some memory pattern $\xi^\mu$ (w.l.g., $\xi^{m+1}$), and letting each bit $x_i$ be either $\xi_i^\mu$ or $-\xi_i^\mu$ with probabilities $\alpha$ and $(1 - \alpha)$ respectively, where $\alpha > 0.5$. To analyze this HN, we use some tight approximations to the binomial distribution. Due to space considerations, their proofs are omitted.

**Lemma 1.**
Let $X \sim Bin(n, p)$. If $x_n$ are integers such that $lim_{n \to \infty} \frac{x_n}{n} = \beta \in (p, 1)$, then

$$P(X \geq x_n) \approx \frac{1 - p}{(1 - \frac{p}{\beta})\sqrt{2\pi n \beta(1 - \beta)}} \exp\{-n[\beta \ln \frac{\beta}{p} + (1 - \beta) \ln \frac{1 - \beta}{1 - p}]\} \quad (1)$$

in the sense that the ratio between LHS and RHS converges to 1 as $n \to \infty$. For the special case $p = \frac{1}{2}$, let $G(\beta) = \ln 2 + \beta \ln \beta + (1 - \beta) \ln(1 - \beta)$, then

$$P(X \geq x_n) \approx \frac{\exp\{-nG(\beta)\}}{(2 - \frac{1}{\beta})\sqrt{2\pi n \beta(1 - \beta)}} \; . \quad (2)$$

**Lemma 2.**
Let $X_i \sim Bin(n, \frac{1}{2})$ be independent, $\gamma \in (0, 1)$, and let $x_n$ be as in Lemma 1. If

$$m = (2 - \frac{1}{\beta})\sqrt{2\pi n \beta(1 - \beta)} \left( \ln \frac{1}{\gamma} \right) e^{nG(\beta)}, \quad (3)$$

then

$$P(max(X_1, X_2, \cdots, X_m) < x_n) \approx \gamma \quad (4)$$

**Lemma 3.**
Let $Y \sim Bin(n, \alpha)$ with $\alpha > \frac{1}{2}$, let $(X_i)$ and $\gamma$ be as in Lemma 2, and let $\eta \in (0, 1)$. Let $x_n$ be the integer closest to $n\beta$, where

$$\beta = \alpha - \sqrt{\frac{\alpha(1 - \alpha)}{n}} z_\eta - \frac{1}{2n} \quad (5)$$

and $z_\eta$ is the $\eta$ - quantile of the standard normal distribution, i.e.,

$$\eta = \frac{1}{\sqrt{2\pi}} \int_{-\infty}^{z_\eta} e^{-x^2/2} dx \qquad (6)$$

Then, if $Y$ and $(X_i)$ are independent

$$P(max(X_1, X_2, \cdots, X_m) < Y) \geq P(max(X_1, X_2, \cdots, X_m) < x_n \leq Y) \Rightarrow \gamma\eta \quad (7)$$

as $n \to \infty$, for $m$ as in (3).

Based on the above binomial probability approximations, we can now propose and analyze a $n$-neuron Threshold Hamming Network (THN) that classifies the input patterns with probability of error not exceeding $\epsilon$, when the input vector is generated with an initial bit-similarity $\alpha$: Let $X_j$ be the similarity between the input vector and the $j'th$ memory pattern $(1 \leq j \leq m)$, and let $Y$ be the similarity with the 'correct' memory pattern $\xi^{m+1}$. Choose $\gamma$ and $\eta$ so that $\gamma\eta \geq 1 - \epsilon$, e.g., $\gamma = \eta = \sqrt{1 - \epsilon}$; determine $\beta$ by (5) and $m$ by (3). Discard the WTA subnet, and simply replace the neurons of the memory layer by $m$ neurons having a threshold $x_n$ , the integer closest to $n\beta$. If any memory neuron with similarity at least $x_n$ is declared 'the winner', then, by Lemma 3, the probability of error is at most $\epsilon$, where 'error' may be due to the existence of no winner, wrong winner, or multiple winners.

# 3    The Hamming Network and an Optimal Threshold Hamming Network

We now calculate the choice of the threshold $x_n$ that maximizes the storage capacity $m = m(n, \epsilon, \alpha)$. Let $\phi$ ($\Phi$) denote the standard normal density (cumulative distribution function), and let $r = \phi/(1 - \Phi)$ denote the corresponding failure rate function. Then,

**Lemma 4.**
The optimal proportion between the two error probabilities is

$$\frac{1-\gamma}{1-\eta} \approx \frac{r(z_\eta)}{\sqrt{n\alpha(1-\alpha)}\ln\frac{\beta}{1-\beta}}, \qquad (8)$$

which we will denote by $\delta$.

**Proof:**
Let $M = max(X_1, X_2, \cdots, X_m)$, and let $Y$ denote the similarity with the 'correct' memory pattern, as before. We have seen that $P(M < x) \approx \exp\{-m\frac{\exp\{-nG(\beta)\}}{\sqrt{2\pi n\beta(1-\beta)(2-\frac{1}{\beta})}}\}$. Since $G'(\beta) = \ln\frac{\beta}{(1-\beta)}$, then by Taylor expansion

$$P(M < x) = P(M < x_0 + x - x_0) \approx \exp\{-m\frac{\exp\{-n[G(\beta + \frac{x-x_0}{n})]\}}{\sqrt{2\pi n\beta(1-\beta)}(2-\frac{1}{\beta})}\} \approx$$

$$\exp\{-m\frac{\exp\{-nG(\beta) - (x-x_0)\ln\frac{\beta}{(1-\beta)}\}}{\sqrt{2\pi n\beta(1-\beta)}(2-\frac{1}{\beta})}\} = \gamma^{(\frac{\beta}{1-\beta})^{x_0-x}} \qquad (9)$$

(in accordance with Gnedenko extreme-value distribution of type 1 (Leadbetter et. al. 1983)). Similarly,

$$P(Y < x) = \exp\{\ln P(Y < x_0 + x - x_0)\} \approx$$

$$P(Y < x_0)\exp\{\frac{\phi(z)}{\Phi^*(z)}\frac{x - x_0}{\sqrt{n\alpha(1-\alpha)}}\} = (1 - \eta)\exp\{r(z)\frac{x - x_0}{\sqrt{n\alpha(1-\alpha)}}\} \quad (10)$$

where $\phi$ is the standard normal density function, $\Phi$ is the standard normal cumulative distribution function, $\Phi^* = 1 - \Phi$ and $r = \frac{\phi}{\Phi^*}$ is the corresponding failure rate function. The probability of correct recognition using a threshold $x$ can now be expressed as

$$P(M < x)P(Y \geq x) = \gamma^{(\frac{\beta}{1-\beta})^{x_0 - x}}(1 - (1 - \eta)\exp\{r(z)\frac{x - x_0}{\sqrt{n\alpha(1-\alpha)}}\}) \quad (11)$$

We differentiate expression (11) with respect to $x_0 - x$, and equate the derivative at $x_0 = x$ to zero, to obtain the relation between $\gamma$ and $\eta$ that yields the optimal threshold, i.e., that which maximizes the probability of correct recognition. This yields

$$\gamma = \exp\{-\frac{r(z)}{\sqrt{n\alpha(1-\alpha)}\ln\frac{\beta}{1-\beta}}\frac{1-\eta}{\eta}\} \quad (12)$$

We now approximate

$$1 - \gamma \approx -\ln\gamma \approx \frac{r(z)}{\sqrt{n\alpha(1-\alpha)}\ln\frac{\beta}{1-\beta}}(1 - \eta) \quad (13)$$

and thus the optimal proportion between the two error probabilities is

$$\frac{1-\gamma}{1-\eta} \approx \frac{r(z)}{\sqrt{n\alpha(1-\alpha)}\ln\frac{\beta}{1-\beta}} = \delta. \quad (14)$$

□

Based on Lemma 4, if the desired probability of error is $\epsilon$, we choose

$$\gamma = 1 - \frac{\delta\epsilon}{1+\delta}, \qquad \eta = 1 - \frac{\epsilon}{(1+\delta)} \, . \quad (15)$$

We start with $\gamma = \eta = \sqrt{1-\epsilon}$, obtain $\beta$ from (5) and $\delta$ from (8), and recompute $\eta$ and $\gamma$ from (15). The limiting values of $\beta$ and $\gamma$ in this iterative process give the maximal capacity $m$ and threshold $x_n$.

We now compute the error probability $\epsilon(m, n, \alpha)$ of the original HN (with the WTA subnet) for arbitrary $m, n$ and $\alpha$, and compare it with $\epsilon$.

**Lemma 5.**
For arbitrary $n, \alpha$ and $\epsilon$, let $m, \beta, \gamma, \eta$ and $\delta$ be as calculated above. Then, the probability of error $\epsilon(m, n, \alpha)$ of the HN satisfies

$$\epsilon(m, n, \alpha) \approx \Gamma(1 - \delta)\frac{1 - e^{-\delta\ln\frac{\beta}{1-\beta}}}{\delta\ln\frac{\beta}{1-\beta}}\frac{\delta^\delta}{(1+\delta)^{1+\delta}}\epsilon^{1+\delta} \quad (16)$$

where

$$\Gamma(t) = \int_0^\infty x^{t-1}e^{-x}dx \tag{17}$$

is the Gamma function.

**Proof:**

$$P(Y \leq M) = \sum_x P(Y \leq x)P(M = x) =$$

$$\sum_x P(Y \leq x)[P(M < x+1) - P(M < x)] \approx$$

$$\sum_x P(Y \leq x_0)e^{-\delta(x_0-x)\ln\frac{\beta}{1-\beta}}$$

$$\left[(P(M < x_0))^{(\frac{\beta}{1-\beta})^{x_0-x-1}} - (P(M < x_0))^{(\frac{\beta}{1-\beta})^{x_0-x}}\right] \tag{18}$$

We now approximate this sum by the integral of the summand: let $b = \frac{\beta}{1-\beta}$ and $c = \delta\ln\frac{\beta}{1-\beta}$. We have seen that the probability of incorrect performance of the WTA subnet is equal to

$$P(Y \leq M) \approx$$

$$\sum_x P(Y \leq x_0)e^{-c(x_0-x)}[(P(M < x_0))^{b^{(x_0-x-1)}} - (P(M < x_0))^{b^{(x_0-x)}}] \approx$$

$$(1-\eta)\int_{-\infty}^\infty (\gamma^{b^{y-1}} - \gamma^{b^y})e^{-cy}dy \tag{19}$$

Now we transform variables $t = b^y \ln\frac{1}{\gamma}$ to get the integral in the form

$$e^{-c}(1-\eta)\int_0^\infty (e^{-t} - e^{-bt})(\frac{t}{\ln\frac{1}{\gamma}})^{\frac{-c}{\ln b}}\frac{dt}{t\ln b} = K_1\int_0^\infty (e^{-t} - e^{-bt})t^{-(1+K_2)}dt \tag{20}$$

This is the convergent difference between two divergent Gamma function integrals. We perform integration by parts to obtain a representation as an integral with $t^{-K_2}$ instead of $t^{-(1+K_2)}$ in the integrand. For $0 \leq K_2 < 1$, the corresponding integral converges. The final result is then

$$(1-\eta)\frac{1-e^{-c}}{c}\Gamma(1 - \frac{c}{\ln b})(\ln\frac{1}{\gamma})^{\frac{c}{\ln b}} \tag{21}$$

Hence, we have

$$P(Y \leq M) \approx (1-\eta)\frac{1-e^{-\delta\ln\frac{\beta}{1-\beta}}}{\delta\ln\frac{\beta}{1-\beta}}\Gamma(1-\delta)(\ln\frac{1}{\gamma})^\delta \approx$$

$$\Gamma(1-\delta)\frac{1-e^{-\delta\ln\frac{\beta}{1-\beta}}}{\delta\ln\frac{\beta}{1-\beta}}\frac{(\epsilon\delta)^\delta}{(1+\delta)^{1+\delta}}\epsilon \tag{22}$$

| % error → threshold , $m$ | predicted THN | predicted HN | experimental THN | experimental HN |
|---|---|---|---|---|
| 133 , 145 | 2.46 $(1 - \gamma = 1.03$ $1 - \eta = 1.46)$ | 0.144 | 2.552 $(1 - \gamma = 1.0$ $1 - \eta = 1.552)$ | 0.103 |
| 134 , 346 | 3.4 $(1 - \gamma = 1.37$ $1 - \eta = 2.11)$ | 0.272 | 3.468 $(1 - \gamma = 1.373$ $1 - \eta = 2.168)$ | 0.253 |
| 135 , 825 | 4.714 $(1 - \gamma = 1.776$ $1 - \eta = 2.991)$ | 0.494 | 4.152 $(1 - \gamma = 1.606$ $1 - \eta = 2.576)$ | 0.485 |
| 136 , 1970 | 6.346 $(1 - \gamma = 2.274$ $1 - \eta = 4.167)$ | 0.857 | 6.447 $(1 - \gamma = 2.335$ $1 - \eta = 4.162)$ | 0.863 |

Table 1: The performance of a HN and optimal THN: A comparison between calculated and experimental results ($\alpha = 0.7, n = 210$).

as claimed. Expression (22) is presented as $K(\epsilon, \delta, \beta)\epsilon$, where $K(\epsilon, \delta, \beta)$ is the factor ($\leq 1$) by which the probability of error $\epsilon$ of the THN should be multiplied in order to get the probability of error of the original HN with the WTA subnet. For small $\delta$, $K$ is close to 1, however, as will be seen in the next section, $K$ is typically larger.

## 4   Numerical results

The experimental results presented in table 1 testify to the accuracy of the HN and THN calculations. Figure 1 presents the calculated error probabilities for various values of input similarity $\alpha$ and memory capacity $m$, as a function of the input size $n$. As is evident, the performance of the THN is worse than that of the HN, but due to the exponential growth of $m$, it requires only a minor increment in $n$ to obtain a THN that performs as well as the original HN.

To examine the sensitivity of the THN network to threshold variation, we have fixed $\alpha = 0.7$, $n = 210$, $m = 825$, and let the threshold vary between 132 and 138. As we can see in figure 2, the threshold 135 is indeed optimal, but the performance with threshold values of 134 and 136 is practically identical. The magnitude of the two error types varies considerably with the threshold value, but this variation has no effect on the overall performance near the optimum. These two error probabilities might as well be taken equal to each other.

**Conclusion**   In this paper we analyzed in detail the performance of a Hamming Network and a Threshold Hamming Network. Given a desired storage capacity and performance, we described how to compute the corresponding minimal network size required. The THN drastically reduces the time and connectivity requirements of Hamming Network classifiers.

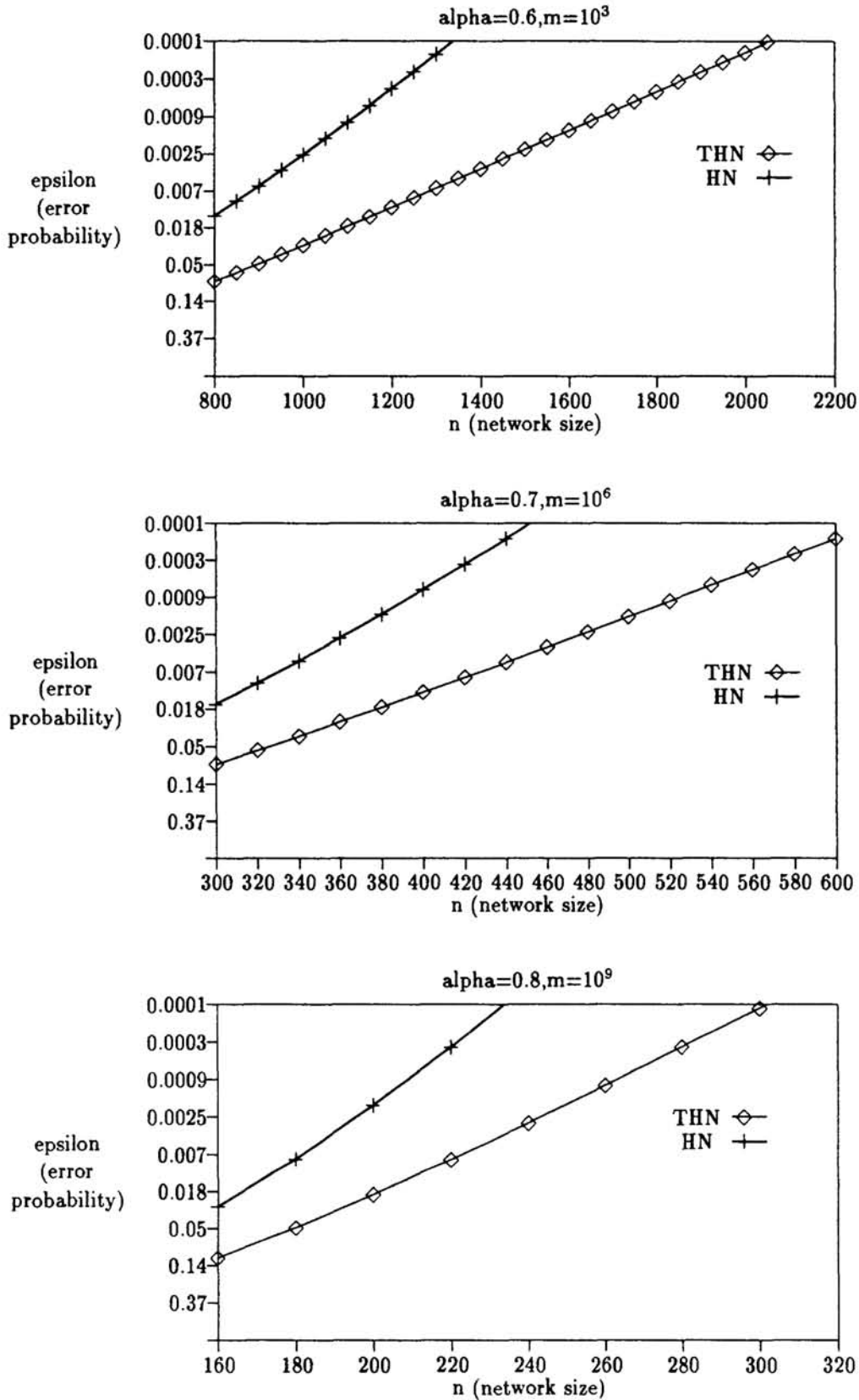

Figure 1: Probability of error as a function of network size: three networks are depicted, displaying the performance at various values of $\alpha$ and $m$. For graphical convenience, we have plotted $\log \frac{1}{\epsilon}$ versus $n$.

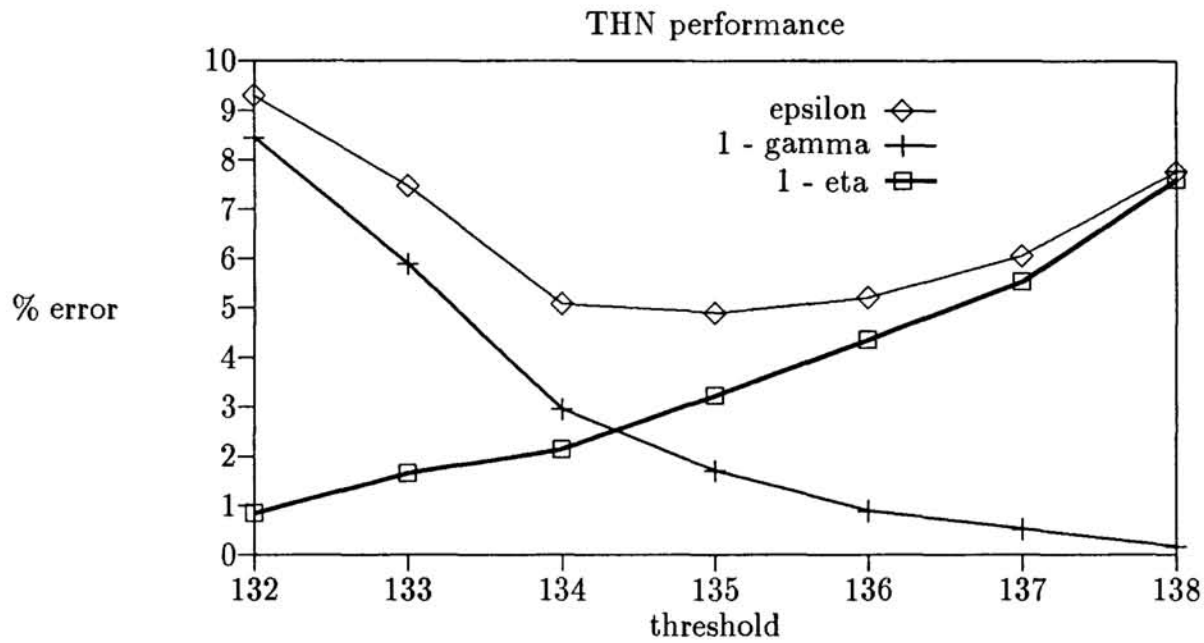

Figure 2: Threshold sensitivity of the THN ($\alpha = 0.7$, $n = 210$, $m = 825$).

# References

[1] K. Steinbuch. Dei lernmatrix. *Kybernetic*, 1:36–45, 1961.

[2] W.K. Taylor. Cortico-thalamic organization and memory. *Proc. of the Royal Society of London B*, 159:466–478, 1964.

[3] R.P. Lippmann, B. Gold, and M.L. Malpass. A comparison of Hamming and Hopfield neural nets for pattern classification. Technical Report TR-769, MIT Lincoln Laboratory, 1987.

[4] E.E. Baum, J. Moody, and F. Wilczek. Internal representations for associative memory. *Biological Cybernetics*, 59:217–228, 1987.

[5] P. Floreen. The convergence of hamming memory networks. *IEEE Trans. on Neural Networks*, 2(4):449–457, 1991.

[6] E. Domany and H. Orland. A maximum overlap neural network for pattern recognition. *Physics Letters A*, 125:32–34, 1987.

[7] M.R. Leadbetter, G. Lindgren, and H. Rootzen. *Extremes and related properties of random sequences and processes*. Springer-Verlag, Berlin-Heidelberg-NewYork, 1983.
